# An Online Algorithm for Maximizing Submodular Functions

**Matthew Streeter**
Google, Inc.
Pittsburgh, PA 15213
mstreeter@google.com

**Daniel Golovin**
Carnegie Mellon University
Pittsburgh, PA 15213
dgolovin@cs.cmu.edu

## Abstract

We present an algorithm for solving a broad class of online resource allocation problems. Our online algorithm can be applied in environments where abstract *jobs* arrive one at a time, and one can complete the jobs by investing time in a number of abstract *activities*, according to some schedule. We assume that the fraction of jobs completed by a schedule is a monotone, submodular function of a set of pairs $(v, \tau)$, where $\tau$ is the time invested in activity $v$. Under this assumption, our online algorithm performs near-optimally according to two natural metrics: (*i*) the fraction of jobs completed within time $T$, for some fixed deadline $T > 0$, and (*ii*) the average time required to complete each job. We evaluate our algorithm experimentally by using it to learn, online, a schedule for allocating CPU time among solvers entered in the 2007 SAT solver competition.

## 1 Introduction

This paper presents an algorithm for solving the following class of online resource allocation problems. We are given as input a finite set $\mathcal{V}$ of *activities*. A pair $(v, \tau) \in \mathcal{V} \times \mathbb{R}_{>0}$ is called an *action*, and represents spending time $\tau$ performing activity $v$. A *schedule* is a sequence of actions. We use $\mathcal{S}$ to denote the set of all schedules. A *job* is a function $f : \mathcal{S} \to [0, 1]$, where for any schedule $S \in \mathcal{S}$, $f(S)$ represents the proportion of some task that is accomplished by performing the sequence of actions $S$. We require that a job $f$ have the following properties (here $\oplus$ is the concatenation operator):

1. (monotonicity) for any schedules $S_1, S_2 \in \mathcal{S}$, we have $f(S_1) \leq f(S_1 \oplus S_2)$ and $f(S_2) \leq f(S_1 \oplus S_2)$

2. (submodularity) for any schedules $S_1, S_2 \in \mathcal{S}$ and any action $a \in \mathcal{V} \times \mathbb{R}_{>0}$, $f_a(S_1 \oplus S_2) \leq f_a(S_1)$, where we define $f_a(S) \equiv f(S \oplus \langle a \rangle) - f(S)$.

We will evaluate schedules in terms of two objectives. The first objective, which we call benefit-maximization, is to maximize $f(S)$ subject to the constraint $\ell(S) \leq T$, for some fixed $T > 0$, where $\ell(S)$ equals the sum of the durations of the actions in $S$. For example if $S = \langle (v_1, 3), (v_2, 3) \rangle$, then $\ell(S) = 6$. The second objective is to minimize the *cost* of a schedule, which we define as

$$c(f, S) = \int_{t=0}^{\infty} 1 - f\left(S_{\langle t \rangle}\right) dt$$

where $S_{\langle t \rangle}$ is the schedule that results from truncating schedule $S$ at time $t$. For example if $S = \langle (v_1, 3), (v_2, 3) \rangle$ then $S_{\langle 5 \rangle} = \langle (v_1, 3), (v_2, 2) \rangle$.[1] One way to interpret this objective is to imagine

that $f(S)$ is the probability that some desired event occurs as a result of performing the actions in $S$. For any non-negative random variable $X$, we have $\mathbb{E}[X] = \int_{t=0}^{\infty} \mathbb{P}[X > t] \, dt$. Thus $c(f, S)$ is the expected time we must wait before the desired event occurs if we execute actions according to the schedule $S$. The following example illustrates these definitions.

**Example 1.** Let each activity $v$ represent a randomized algorithm for solving some decision problem, and let the action $(v, \tau)$ represent running the algorithm (with a fresh random seed) for time $\tau$. Fix some particular instance of the decision problem, and for any schedule $S$, let $f(S)$ be the probability that one (or more) of the runs in the sequence $S$ yields a solution to that instance. So $f(S_{\langle T \rangle})$ is (by definition) the probability that performing the runs in schedule $S$ yields a solution to the problem instance in time $\leq T$, while $c(f, S)$ is the *expected* time that elapses before a solution is obtained. It is clear that $f(S)$ is monotone, because adding runs to the sequence $S$ can only increase the probability that one of the runs is successful. The fact that $f$ is submodular can be seen as follows. For any schedule $S$ and action $a$, $f_a(S)$ equals the probability that action $a$ succeeds after every action in $S$ has failed, which can also be written as $(1 - f(S)) \cdot f(\langle a \rangle)$. This, together with the monotonicity of $f$, implies that for any schedules $S_1, S_2$ and any action $a$, we have $f_a(S_1 \oplus S_2) = (1 - f(S_1 \oplus S_2)) \cdot f(\langle a \rangle) \leq (1 - f(S_1)) \cdot f(\langle a \rangle) = f_a(S_1)$.

In the online setting, an arbitrary sequence $\langle f^{(1)}, f^{(2)}, \ldots, f^{(n)} \rangle$ of jobs arrive one at a time, and we must finish each job (via some schedule) before moving on to the next job. When selecting a schedule $S^{(i)}$ to use to finish job $f^{(i)}$, we have knowledge of the previous jobs $f^{(1)}, f^{(2)}, \ldots, f^{(i-1)}$ but we have no knowledge of $f^{(i)}$ itself or of any subsequent jobs. In this setting we aim to minimize *regret*, which measures the difference between the average cost (or average benefit) of the schedules produced by our online algorithm and that of the best single schedule (in hindsight) for the given sequence of jobs.

## 1.1 Problems that fit into this framework

A number of previously-studied problems can be cast as the task of computing a schedule $S$ that minimizes $c(f, S)$, where $f$ is of the form $f(S) = \frac{1}{n} \sum_{i=1}^{n} \left( 1 - \prod_{(v,\tau) \in S} (1 - p_i(v, \tau)) \right)$. This expression can be interpreted as follows: the job $f$ consists of $n$ subtasks, and $p_i(v, \tau)$ is the probability that investing time $\tau$ in activity $v$ completes the $i^{\text{th}}$ subtask. Thus, $f(S)$ is the expected fraction of subtasks that are finished after performing the sequence of actions $S$. Assuming $p_i(v, \tau)$ is a non-decreasing function of $\tau$ for all $i$ and $v$, it can be shown that any function $f$ of this form is monotone and submodular. PIPELINED SET COVER [11, 15] can be defined as the special case in which for each activity $v$ there is an associated time $\tau_v$, and $p_i(v, \tau) = 1$ if $\tau \geq \tau_v$ and $p_i(v, \tau) = 0$ otherwise. MIN-SUM SET COVER [7] is the special case in which, additionally, $\tau_v = 1$ or $\tau_v = \infty$ for all $v \in \mathcal{V}$. The problem of constructing efficient sequences of trials [5] corresponds to the case in which we are given a matrix $q$, and $p_i(v, \tau) = q_{v,i}$ if $\tau \geq 1$ and $p_i(v, \tau) = 0$ otherwise.

The problem of maximizing $f(S_{\langle T \rangle})$ is a slight generalization of the problem of maximizing a monotone submodular set function subject to a knapsack constraint [14, 20] (which in turn generalizes BUDGETED MAXIMUM COVERAGE [12], which generalizes MAX $k$-COVERAGE [16]). The only difference between the two problems is that, in the latter problem, $f(S)$ may only depend on the *set* of actions in the sequence $S$, and not on the order in which the actions appear.

## 1.2 Applications

We now discuss three applications, the first of which is the focus of our experiments in §5.

*1. Online algorithm portfolio design.* An *algorithm portfolio* [9] is a schedule for interleaving the execution of multiple (randomized) algorithms and periodically restarting them with a fresh random seed. Previous work has shown that combining multiple heuristics for NP-hard problems into a portfolio can dramatically reduce average-case running time [8, 9, 19]. In particular, algorithms based on chronological backtracking often exhibit heavy-tailed run length distributions, and periodically restarting them with a fresh random seed can reduce the mean running time by orders of magnitude [8]. As illustrated in Example 1, our algorithms can be used to learn an effective algorithm portfolio online, in the course of solving a sequence of problem instances.

*2. Database query processing.* In database query processing, one must extract all the records in a database that satisfy every predicate in a list of one or more predicates (the conjunction of predicates comprises the query). To process the query, each record is evaluated against the predicates one at a time until the record either fails to satisfy some predicate (in which case it does not match the query) or all predicates have been examined. The order in which the predicates are examined affects the time required to process the query. Munagala *et al.* [15] introduced and studied a problem called PIPELINED SET COVER (discussed in §1.1), which entails finding an evaluation order for the predicates that minimizes the average time required to process a record. Our work addresses the online version of this problem, which arises naturally in practice.

*3. Sensor placement.* Sensor placement is the task of assigning locations to a set of sensors so as to maximize the value of the information obtained (e.g., to maximize the number of intrusions that are detected by the sensors). Many sensor placement problems can be optimally solved by maximizing a monotone submodular set function subject to a knapsack constraint [13], a special case of our benefit-maximization problem (see §1.1). Our online algorithms could be used to select sensor placements when the same set of sensors is repeatedly deployed in an unknown or adversarial environment.

### 1.3   Summary of results

We first consider the offline variant of our problem. As an immediate consequence of existing results [6, 7], we find that, for any $\epsilon > 0$, (*i*) achieving an approximation ratio of $4 - \epsilon$ for the cost-minimization problem is NP-hard and (*ii*) achieving an approximation ratio of $1 - \frac{1}{e} + \epsilon$ for the benefit-maximization problem is NP-hard. We then present a greedy approximation algorithm that simultaneously achieves the optimal approximation ratios (of $4$ and $1 - \frac{1}{e}$) for these two problems, building on and generalizing previous work on special cases of these two problems [7, 20].

In the online setting we provide an online algorithm whose worst-case performance guarantees approach those of the offline greedy approximation algorithm asymptotically (as the number of jobs approaches infinity). We then show how to modify our online algorithm for use in several different "bandit" feedback settings. Finally, we prove information-theoretic lower bounds on regret. We conclude with an experimental evaluation.

## 2   Related Work

As discussed in §1.1, the offline cost-minimization problem considered here generalizes MIN-SUM SET COVER [7], PIPELINED SET COVER [11, 15], and the problem of constructing efficient sequences of trials [5]. Several of these problems have been considered in the online setting. Munagala *et al.* [15] gave an online algorithm for PIPELINED SET COVER that is asymptotically $O\left(\log |\mathcal{V}|\right)$-competitive. Babu *et al.* [3] and Kaplan *et al.* [11] gave online algorithms for PIPELINED SET COVER that are asymptotically 4-competitive, but only in the special case where the jobs are drawn independently at random from a fixed probability distribution (whereas our online algorithm is asymptotically 4-competitive on an arbitrary sequence of jobs).

Our offline benefit-maximization problem generalizes the problem of maximizing a monotone submodular set function subject to a knapsack constraint. Previous work gave offline greedy approximation algorithms for this problem [14, 20], which generalized earlier algorithms for BUDGETED MAXIMUM COVERAGE [12] and MAX $k$-COVERAGE [16]. To our knowledge, none of these problems have previously been studied in an online setting. Note that our problem is quite different from online set covering problems (e.g., [1]) that require one to construct a *single* collection of sets that covers each element in a sequence of elements that arrive online.

In this paper we convert a specific greedy approximation algorithm into an online algorithm. Recently, Kakade *et al.* [10] gave a generic procedure for converting an $\alpha$-approximation algorithm into an online algorithm that is asymptotically $\alpha$-competitive. Their algorithm applies to linear optimization problems, but not to the non-linear problems we consider here.

Independently of us, Radlinkski *et al.* [17] developed a no-regret algorithm for the online version of MAX $k$-COVERAGE, and applied it to online ranking. As it turns out, their algorithm is a special case of the algorithm $\mathbf{OG_{unit}}$ that we present in §4.1.

# 3 Offline Greedy Algorithm

In the offline setting, we are given as input a job $f : \mathcal{S} \to [0, 1]$. Our goal is to compute a schedule $S$ that achieves one of two objectives, either minimizing the cost $c(f, S)$ or maximizing $f(S)$ subject to the constraint $\ell(S) \leq T$.[2] As already mentioned, this offline problem generalizes MIN-SUM SET COVER under the former objective and generalizes MAX $k$-COVERAGE under the latter objective, which implies the following computational complexity result [6, 7].

**Theorem 1.** *For any $\epsilon > 0$, achieving a $4 - \epsilon$ (resp. $1 - \frac{1}{e} + \epsilon$) approximation ratio for the cost-minimization (resp. benefit-maximization) problem is NP-hard.*

We now consider an arbitrary schedule $G$, whose $j^{\text{th}}$ action is $g_j = (v_j, \tau_j)$. Let $s_j = \frac{f_{g_j}(G_j)}{\tau_j}$, where $G_j = g_1 g_2 \cdots g_{j-1}$, and let $\gamma_j = \max_{(v,\tau): v \in \mathcal{V}, \tau \in \mathbb{R}_{>0}} \frac{f_{(v,\tau)}(G_j)}{\tau} - s_j$. We will prove bounds on the performance of $G$ in terms of the $\gamma_j$ values. Note that we can ensure $\gamma_j = 0 \; \forall j$ by greedily choosing $g_j = \arg\max_{(v,\tau): v \in \mathcal{V}, \tau \in \mathbb{R}_{>0}} \frac{f_{(v,\tau)}(G_j)}{\tau}$ (i.e., greedily appending actions to the schedule so as to maximize the resulting increase in $f$ per unit time). A key property is stated in the following lemma, which follows from the submodularity assumption (for the proof, see [18]).

**Lemma 1.** *For any schedule $S$, any positive integer $j$, and any $t > 0$, $f(S_{\leq t}) \leq f(G_j) + t \cdot (s_j + \gamma_j)$.*

Using Lemma 1, together with a geometric proof technique developed in [7], we now show that the greedy algorithm achieves the optimal approximation ratio for the cost-minimization problem.

**Theorem 2.** *Let $S^* = \arg\min_{S \in \mathcal{S}} c(f, S)$. If $\gamma_j = 0 \; \forall j$, then $c(f, G) \leq 4 \cdot c(f, S^*)$. More generally, let $L$ be a positive integer, and let $T = \sum_{j=1}^{L} \tau_j$. For any schedule $S$, define $c^T(f, S) \triangleq \int_{t=0}^{T} 1 - f(S_{\leq t}) \, dt$. Then $c^T(f, G) \leq 4 \cdot c(f, S^*) + \sum_{j=1}^{L} E_j \tau_j$, where $E_j = \sum_{l < j} \gamma_l \tau_l$.*

*Proof.* We consider the special case $\gamma_j = 0 \; \forall j$; for the full proof see [18]. Let $R_j = 1 - f(G_j)$; let $x_j = \frac{R_j}{2s_j}$; let $y_j = \frac{R_j}{2}$; and let $h(x) = 1 - f(S^*_{\leq x})$. By Lemma 1, $h(x_j) \geq R_j - \frac{R_j}{2} = y_j$. The monotonicity of $f$ implies that $h(x)$ is non-increasing and also that the sequence $y_1, y_2, \ldots$ is non-increasing. These facts imply that $\int_{x=0}^{\infty} h(x) \, dx \geq \sum_{j \geq 1} x_j (y_j - y_{j+1})$ (see Figure 1). The left hand side equals $c(f, S^*)$, and, using the fact that $s_j = \frac{R_j - R_{j+1}}{\tau_j}$, the right hand side simplifies to $\frac{1}{4} \sum_{j \geq 1} R_j \tau_j = \frac{1}{4} c(f, G)$, proving $c(f, G) \leq 4 \cdot c(f, S^*)$. $\square$

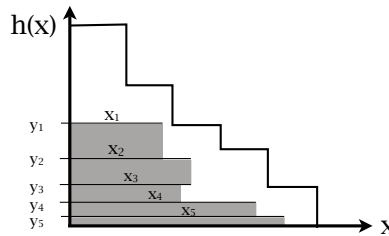

Figure 1: An illustration of the inequality $\int_{x=0}^{\infty} h(x) \, dx \geq \sum_{j \geq 1} x_j (y_j - y_{j+1})$.

The greedy algorithm also achieves the optimal approximation ratio for the benefit-maximization problem, as can be shown using arguments similar to the ones in [14, 20]; see [18] for details.

**Theorem 3.** *Let $L$ be a positive integer, and let $T = \sum_{j=1}^{L} \tau_j$. Then $f(G_{\leq T}) > (1 - \frac{1}{e}) \max_{S \in \mathcal{S}} f(S_{\leq T}) - \sum_{j=1}^{L} \gamma_j \tau_j$.*

# 4 Online Greedy Algorithm

In the online setting we are fed, one at a time, a sequence $\langle f^{(1)}, f^{(2)}, \ldots, f^{(n)} \rangle$ of jobs. Prior to receiving job $f^{(i)}$, we must specify a schedule $S^{(i)}$. We then receive complete access to the function $f^{(i)}$.

We measure performance using two different notions of regret. For the cost-minimization objective, we define $R_{\text{cost}} = \frac{1}{n} \sum_{i=1}^{n} c^T \left( S^{(i)}, f^{(i)} \right) - 4 \cdot \min_{S \in \mathcal{S}} \left\{ \frac{1}{n} \sum_{i=1}^{n} c \left( S, f^{(i)} \right) \right\}$, for some fixed $T > 0$. Here for any schedule $S$ and job $f$, we define $c^T (S, f) = \int_{t=0}^{T} 1 - f \left( S_{\langle t \rangle} \right) \, dt$ to be the value of $c(S, f)$ when the integral is truncated at time $T$. Some form of truncation is necessary because $c \left( S^{(i)}, f^{(i)} \right)$ could be infinite, and without bounding it we could not prove any finite bound on regret (our regret bounds will be stated as a function of $T$). For the benefit-maximization objective, we define $R_{\text{benefit}} = \left( 1 - \frac{1}{e} \right) \max_{S \in \mathcal{S}} \left\{ \frac{1}{n} \sum_{i=1}^{n} f^{(i)} \left( S_{\langle T \rangle} \right) \right\} - \frac{1}{n} \sum_{i=1}^{n} f^{(i)} \left( S^{(i)} \right)$. Here we require that for each $i$, $\mathbb{E} \left[ \ell \left( S^{(i)} \right) \right] = T$, where the expectation is over the online algorithm's random bits. That is, we allow the online algorithm to treat $T$ as a budget in expectation, rather than a hard budget.

Our goal is to bound the worst-case expected values of $R_{\text{cost}}$ and $R_{\text{benefit}}$. For simplicity, we consider the *oblivious adversary model*, in which the sequence of jobs is fixed in advance and does not change in response to the decisions made by our online algorithm. We confine our attention to schedules that consist of actions that come from some finite set $\mathcal{A}$, and assume that the actions in $\mathcal{A}$ have integer durations (i.e. $\mathcal{A} \subseteq \mathcal{V} \times \mathbb{Z}_{>0}$).

## 4.1 Unit-cost actions

In the special case in which each action takes unit time (i.e., $\mathcal{A} \subseteq \mathcal{V} \times \{1\}$), our online algorithm $\mathbf{OG_{unit}}$ is very simple. $\mathbf{OG_{unit}}$ runs $T$ action-selection algorithms, $\mathcal{E}_1, \mathcal{E}_2, \ldots, \mathcal{E}_T$, where $T$ is the number of time steps for which our schedule is defined. The intent is that each action-selection algorithm is a no-regret algorithm such as randomized weighted majority (WMR) [4], which selects actions so as to maximize payoffs associated with the actions. Just before job $f^{(i)}$ arrives, each action-selection algorithm $\mathcal{E}_t$ selects an action $a_t^i$. The schedule used by $\mathbf{OG_{unit}}$ on job $f^{(i)}$ is $S^{(i)} = \langle a_1^i, a_2^i, \ldots, a_T^i \rangle$. The payoff that $\mathcal{E}_t$ associates with action $a$ is $f_a^{(i)} \left( S_{\langle t-1 \rangle}^{(i)} \right)$.

**Theorem 4.** *Algorithm* $\mathbf{OG_{unit}}$ *has* $\mathbb{E} \left[ R_{\text{benefit}} \right] = O \left( \sqrt{\frac{T}{n} \ln |\mathcal{A}|} \right)$ *and* $\mathbb{E} \left[ R_{\text{cost}} \right] = O \left( T \sqrt{\frac{T}{n} \ln |\mathcal{A}|} \right)$ *in the worst case, when WMR [4] is the subroutine action-selection algorithm.*

*Proof.* We will view $\mathbf{OG_{unit}}$ as producing an approximate version of the offline greedy schedule for the job $f = \frac{1}{n} \sum_{i=1}^{n} f^{(i)}$. First, view the sequence of actions selected by $\mathcal{E}_t$ as a single *meta-action* $\tilde{a}_t$, and extend the domain of each $f^{(i)}$ to include the meta-actions by defining $f^{(i)}(S \oplus \langle \tilde{a}_t \rangle) = f^{(i)}(S \oplus \langle a_t^i \rangle)$ for all $S \in \mathcal{S}$ (note each $f^{(i)}$ remains monotone and submodular). Thus, the online algorithm produces a single schedule $\tilde{S} = \langle \tilde{a}_1, \tilde{a}_2, \ldots, \tilde{a}_T \rangle$ for all $i$. Let $r_t$ be the regret experienced by action-selection algorithm $\mathcal{E}_t$. By construction, $r_t = \max_{a \in \mathcal{A}} \left\{ f_a \left( \tilde{S}_{\langle t-1 \rangle} \right) \right\} - f_{\tilde{a}_t} \left( \tilde{S}_{\langle t-1 \rangle} \right)$. Thus $\mathbf{OG_{unit}}$ behaves exactly like the greedy schedule $G$ for the function $f$, with $\epsilon_t = r_t$. Thus, Theorem 3 implies that $R_{\text{benefit}} \leq \sum_{t=1}^{T} r_t \equiv R$. Similarly, Theorem 2 implies that $R_{\text{cost}} \leq TR$.

To complete the analysis, it remains to bound $\mathbb{E}[R]$. WMR has worst-case expected regret $O \left( \frac{1}{n} \sqrt{G_{\max} \ln |\mathcal{A}|} \right)$, where $G_{\max}$ is the maximum sum of payoffs payoff for any single action.[3] Because each payoff is at most 1 and there are $n$ rounds, $G_{\max} \leq n$, so a trivial bound is $\mathbb{E}[R] = O \left( T \sqrt{\frac{1}{n} \ln |\mathcal{A}|} \right)$. In fact, the worst case is when $G_{\max} = \Theta \left( \frac{n}{T} \right)$ for all $T$ action-selection algorithms, leading to an improved bound of $\mathbb{E}[R] = O \left( \sqrt{\frac{T}{n} \ln |\mathcal{A}|} \right)$ (for details see [18]), which completes the proof. $\qquad \square$

## 4.2 From unit-cost actions to arbitrary actions

In this section we generalize the online greedy algorithm presented in the previous section to accommodate actions with arbitrary durations. Like $\mathbf{OG_{unit}}$, our generalized algorithm $\mathbf{OG}$ makes use of a series of action-selection algorithms $\mathcal{E}_1, \mathcal{E}_2, \ldots, \mathcal{E}_L$ (for $L$ to be determined). On each round $i$, $\mathbf{OG}$ constructs a schedule $S^{(i)}$ as follows: for $t = 1, 2, \ldots, L$, it uses $\mathcal{E}_t$ to choose an action $a_t^i = (v, \tau) \in \mathcal{A}$, and appends this action to $S^{(i)}$ with probability $\frac{1}{\tau}$. Let $S_t^{(i)}$ denote the schedule that results from the first $t$ steps of this process (so $S_t^{(i)}$ contains between 0 and $t$ actions). The payoff that $\mathcal{E}_t$ associates with an action $a = (v, \tau)$ equals $\frac{1}{\tau} f_a(S_{t-1}^{(i)})$ (i.e., the increase in $f$ per unit time that would have resulted from appending $a$ to the schedule-under-construction).

As in the previous section, we view each action-selection algorithm $\mathcal{E}_t$ as selecting a single *meta-action* $\tilde{a}_t$. We extend the domain of each $f^{(i)}$ to include the meta-actions by defining $f^{(i)}(S \oplus \langle \tilde{a}_t \rangle) = f^{(i)}(S \oplus \langle a_t^i \rangle)$ if $a_t^i$ was appended to $S^{(i)}$, and $f^{(i)}(S \oplus \langle \tilde{a}_t \rangle) = f^{(i)}(S)$ otherwise. Thus, the online algorithm produces a single schedule $\tilde{S} = \langle \tilde{a}_1, \tilde{a}_2, \ldots, \tilde{a}_L \rangle$ for all $i$. Note that each $f^{(i)}$ remains monotone and submodular.

For the purposes of analysis, we will imagine that each meta-action $\tilde{a}_t$ *always* takes unit time (whereas in fact, $\tilde{a}_t$ takes unit time per job in expectation). We show later that this assumption does not invalidate any of our arguments.

Let $f = \frac{1}{n} \sum_{i=1}^{n} f^{(i)}$, and let $\tilde{S}_t = \langle \tilde{a}_1, \tilde{a}_2, \ldots, \tilde{a}_t \rangle$. Thus $\tilde{S}$ can be viewed as a version of the greedy schedule from §3, with $\epsilon_t = \max_{(v,\tau) \in \mathcal{A}} \left\{ \frac{1}{\tau} \left( f_{(v,\tau)}(\tilde{S}_{t-1}) \right) \right\} - \left( f_{\tilde{a}_t}(\tilde{S}_{t-1}) \right)$, where we are using the assumption that $\tilde{a}_t$ takes unit time. Let $r_t$ be the regret experienced by $\mathcal{E}_t$. Although $r_t \neq \epsilon_t$ in general, the two quantities are equal in expectation (proof omitted).

**Lemma 2.** $\mathbb{E}\left[\epsilon_t\right] = \mathbb{E}\left[r_t\right]$.

We now prove a bound on $\mathbb{E}\left[R_{\text{benefit}}\right]$. Because each $f^{(i)}$ is monotone and submodular, $f$ is monotone and submodular as well, so the greedy schedule's approximation guarantees apply to $f$. In particular, by Theorem 3, we have $R_{\text{benefit}} \leq \sum_{t=1}^{T} \epsilon_t$. Thus by Lemma 2, $\mathbb{E}\left[R_{\text{benefit}}\right] \leq \mathbb{E}\left[R\right]$, where $R = \sum_{t=1}^{T} r_t$.

To bound $\mathbb{E}\left[R_{\text{benefit}}\right]$, it remains to justify the assumption that each meta-action $\tilde{a}_t$ always takes unit time. First, note that the value of the objective function $f(\tilde{S})$ is independent of how long each meta-action $\tilde{a}_t$ takes. Thus, the only potential danger is that in making this assumption we have overlooked a constraint violation of the form $\mathbb{E}\left[\ell\left(S^{(i)}\right)\right] \neq T$. But by construction, $\mathbb{E}\left[\ell\left(S^{(i)}\right)\right] = L$ for each $i$, regardless of what actions are chosen by each action-selection algorithm. Thus if we set $L = T$ there is no constraint violation. Combining the bound on $\mathbb{E}\left[R\right]$ stated in the proof of Theorem 4 with the fact that $\mathbb{E}\left[R_{\text{benefit}}\right] \leq \mathbb{E}\left[R\right]$ yields the following theorem.

**Theorem 5.** *Algorithm* $\mathbf{OG}$, *run with input* $L = T$, *has* $\mathbb{E}\left[R_{\text{benefit}}\right] \leq \mathbb{E}\left[R\right]$. *If WMR [4] is used as the subroutine action-selection algorithm, then* $\mathbb{E}\left[R\right] = O\left(\sqrt{\frac{T}{n} \ln |\mathcal{A}|}\right)$.

The argument bounding $\mathbb{E}\left[R_{\text{cost}}\right]$ is similar, although somewhat more involved (for details, see [18]). One additional complication is that $\ell\left(S^{(i)}\right)$ is now a random variable, whereas in the definition of $R_{\text{cost}}$ the cost of a schedule is always calculated up to time $T$. This can be addressed by making the probability that $\ell\left(S^{(i)}\right) < T$ sufficiently small, which can be done by setting $L \gg T$ and applying concentration of measure inequalities. However, $\mathbb{E}\left[R\right]$ grows as a function of $L$, so we do not want to make $L$ too large. The (approximately) best bound is obtained by setting $L = T \ln n$.

**Theorem 6.** *Algorithm* $\mathbf{OG}$, *run with input* $L = T \ln n$, *has* $\mathbb{E}\left[R_{\text{cost}}\right] = O(T \ln n \cdot \mathbb{E}\left[R\right] + \frac{T}{\sqrt{n}})$. *In particular,* $\mathbb{E}\left[R_{\text{cost}}\right] = O\left((\ln n)^{\frac{3}{2}} T \sqrt{\frac{T}{n} \ln |\mathcal{A}|}\right)$ *if WMR [4] is used as the subroutine action-selection algorithm.*

### 4.3 Dealing with limited feedback

Thus far we have assumed that, after specifying a schedule $S^{(i)}$, the online algorithm receives complete access to the job $f^{(i)}$. We now consider three more limited feedback settings that may arise in practice. In the *priced feedback model*, to receive access to $f^{(i)}$ we must pay a price $C$, which is added to our regret. In the *partially transparent feedback model*, we only observe $f^{(i)}\left(S^{(i)}_{\langle t \rangle}\right)$ for each $t > 0$. In the *opaque feedback model*, we only observe $f^{(i)}\left(S^{(i)}\right)$.

The priced and partially transparent feedback models arise naturally in the case where action $(v, \tau)$ represents running a deterministic algorithm $v$ for $\tau$ time units, and $f(S) = 1$ if some action in $S$ yields a solution to some particular problem instance, and $f(S) = 0$ otherwise. If we execute a schedule $S$ and halt as soon as some action yields a solution, we obtain exactly the information that is revealed in the partially transparent model. Alternatively, running each algorithm $v$ until it returns a solution would completely reveal the function $f^{(i)}$, but incurs a computational cost, as reflected in the priced feedback model.

Algorithm **OG** can be adapted to work in each of these three feedback settings; see [18] for the specific bounds. In all cases, the high-level idea is to replace the unknown quantities used by **OG** with (unbiased) estimates of those quantities. This technique has been used in a number of online algorithms (e.g., see [2]).

### 4.4 Lower bounds on regret

We now state lower bounds on regret; for the proofs see the full paper [18]. Our proofs have the same high-level structure as that of the lower bound given in [4], in that we define a distribution over jobs that allows any online algorithm's expected performance to be easily bounded, and then prove a bound on the expected performance of the best schedule in hindsight. The upper bounds in Theorem 4 match the lower bounds in Theorem 7 up to logarithmic factors, although the latter apply to standard regret as opposed to $R_{\text{benefit}}$ and $R_{\text{cost}}$ (which include factors of $1 - \frac{1}{e}$ and 4).

**Theorem 7.** *Let* $X = \sqrt{\frac{T}{n} \ln \frac{|\mathcal{V}|}{T}}$. *Then any online algorithm has worst-case expected regret* $\Omega\left(X\right)$ *(resp.* $\Omega\left(TX\right)$*) for the online benefit-maximization (resp. cost-minimization) problem.*

## 5 Experimental Evaluation on SAT 2007 Competition Data

The annual SAT solver competition (`www.satcompetition.org`) is designed to encourage the development of efficient Boolean satisfiability solvers, which are used as subroutines in state-of-the-art model checkers, theorem provers, and planners. The competition consists of running each submitted solver on a number of benchmark instances, with a per-instance time limit. Solvers are ranked according to the instances they solve within each of three instance categories: *industrial*, *random*, and *hand-crafted*.

We evaluated the online algorithm **OG** by using it to combine solvers from the 2007 SAT solver competition. To do so, we used data available on the competition web site to construct a matrix $X$, where $X_{i,j}$ is the time that the $j^{\text{th}}$ solver required on the $i^{\text{th}}$ benchmark instance. We used this data to determine whether or not a given schedule would solve an instance within the time limit $T$ (schedule $S$ solves instance $i$ if and only if, for some $j$, $S_{\langle T \rangle}$ contains an action $(h_j, \tau)$ with $\tau \geq X_{i,j}$). As illustrated in Example 1, the task of maximizing the number of instances solved within the time limit, in an online setting in which a sequence of instances must be solved one at a time, is an instance of our online problem (under the benefit-maximization objective).

Within each instance category, we compared **OG** to the offline greedy schedule, to the individual solver that solved the most instances within the time limit, and to a schedule that ran each solver in parallel at equal strength. For these experiments, we ran **OG** in the full-information feedback model, after finding that the number of benchmark instances was too small for **OG** to be effective in the limited feedback models. Table 1 summarizes the results. In each category, the offline greedy schedule and the online greedy algorithm outperform all solvers entered in the competition as well as the naïve parallel schedule.

Table 1: Number of benchmark instances solved within the time limit.

| Category | Offline greedy | Online greedy | Parallel schedule | Top solver |
|---|---|---|---|---|
| *Industrial* | 147 | 149 | 132 | 139 |
| *Random* | 350 | 347 | 302 | 257 |
| *Hand-crafted* | 114 | 107 | 95 | 98 |

## Footnotes

[1]More formally, if $S = \langle a_1, a_2, \ldots \rangle$, where $a_i = (v_i, \tau_i)$, then $S_{\langle t \rangle} = \langle a_1, a_2, \ldots, a_{k-1}, a_k, (v_{k+1}, \tau') \rangle$, where $k$ is the largest integer such that $\sum_{i=1}^{k} \tau_i < t$ and $\tau' = t - \sum_{i=1}^{k} \tau_i$.

[2]Given a set of jobs $f^{(1)}, f^{(2)}, \ldots, f^{(n)}$, we can optimize the average schedule cost (or benefit) simply by applying our offline algorithm to the job $f = \frac{1}{n} \sum_{i=1}^{n} f^{(i)}$ (since any convex combination of jobs is a job).

[3]This bound requires $G_{\max}$ to be known in advance; however, the same guarantee can be achieved by guessing a value of $G_{\max}$ and doubling the guess whenever it is proven wrong.

# References

[1] Noga Alon, Baruch Awerbuch, and Yossi Azar. The online set cover problem. In *Proceedings of the* $35^{th}$ *STOC*, pages 100–105, 2003.

[2] Peter Auer, Nicolò Cesa-Bianchi, Yoav Freund, and Robert E. Schapire. The nonstochastic multiarmed bandit problem. *SIAM Journal on Computing*, 32(1):48–77, 2002.

[3] Shivnath Babu, Rajeev Motwani, Kamesh Munagala, Itaru Nishizawa, and Jennifer Widom. Adaptive ordering of pipelined stream filters. In *Proc. Intl. Conf. on Management of Data*, pages 407–418, 2004.

[4] Nicolò Cesa-Bianchi, Yoav Freund, David Haussler, David Helmbold, Robert Schapire, and Manfred Warmuth. How to use expert advice. *Journal of the ACM*, 44(3):427–485, 1997.

[5] Edith Cohen, Amos Fiat, and Haim Kaplan. Efficient sequences of trials. In *Proceedings of the* $14^{th}$ *SODA*, pages 737–746, 2003.

[6] Uriel Feige. A threshold of $\ln n$ for approximating set cover. *Journal of the ACM*, 45(4):634–652, 1998.

[7] Uriel Feige, László Lovász, and Prasad Tetali. Approximating min sum set cover. *Algorithmica*, 40(4):219–234, 2004.

[8] Carla P. Gomes and Bart Selman. Algorithm portfolios. *Artificial Intelligence*, 126:43–62, 2001.

[9] Bernardo A. Huberman, Rajan M. Lukose, and Tad Hogg. An economics approach to hard computational problems. *Science*, 275:51–54, 1997.

[10] Sham Kakade, Adam Kalai, and Katrina Ligett. Playing games with approximation algorithms. In *Proceedings of the* $39^{th}$ *STOC*, pages 546–555, 2007.

[11] Haim Kaplan, Eyal Kushilevitz, and Yishay Mansour. Learning with attribute costs. In *Proceedings of the* $37^{th}$ *STOC*, pages 356–365, 2005.

[12] Samir Khuller, Anna Moss, and Joseph (Seffi) Naor. The budgeted maximum coverage problem. *Information Processing Letters*, 70(1):39–45, 1999.

[13] Andreas Krause and Carlos Guestrin. Near-optimal nonmyopic value of information in graphical models. In *Proceedings of the* $21^{st}$ *UAI*, pages 324–331, 2005.

[14] Andreas Krause and Carlos Guestrin. A note on the budgeted maximization of submodular functions. Technical Report CMU-CALD-05-103, Carnegie Mellon University, 2005.

[15] Kamesh Munagala, Shivnath Babu, Rajeev Motwani, Jennifer Widom, and Eiter Thomas. The pipelined set cover problem. In *Proc. Intl. Conf. on Database Theory*, pages 83–98, 2005.

[16] G. L. Nemhauser, L. A. Wolsey, and M. L. Fisher. An analysis of approximations for maximizing submodular set functions. *Mathematical Programming*, 14(1):265–294, 1978.

[17] Filip Radlinski, Robert Kleinberg, and Thorsten Joachims. Learning diverse rankings with multi-armed bandits. In *Proceedings of the* $25^{th}$ *ICML*, pages 784–791, 2008.

[18] Matthew Streeter and Daniel Golovin. An online algorithm for maximizing submodular functions. Technical Report CMU-CS-07-171, Carnegie Mellon University, 2007.

[19] Matthew Streeter, Daniel Golovin, and Stephen F. Smith. Combining multiple heuristics online. In *Proceedings of the* $22^{nd}$ *AAAI*, pages 1197–1203, 2007.

[20] Maxim Sviridenko. A note on maximizing a submodular set function subject to a knapsack constraint. *Operations Research Letters*, 32:41–43, 2004.

